# HIERARCHICAL LEARNING CONTROL -

# AN APPROACH WITH NEURON-LIKE ASSOCIATIVE MEMORIES

E. Ersü
ISRA Systemtechnik GmbH, Schöfferstr. 15, D-6100 Darmstadt, FRG

H. Tolle
TH Darmstadt, Institut für Regelungstechnik,
Schloßgraben 1, D-6100 Darmstadt, FRG

## ABSTRACT

Advances in brain theory need two complementary approaches: Analytical investigations by in situ measurements and as well synthetic modelling supported by computer simulations to generate suggestive hypothesis on purposeful structures in the neural tissue. In this paper research of the second line is described: Starting from a neurophysiologically inspired model of stimulus-response (S-R) and/or associative memorization and a psychologically motivated ministructure for basic control tasks, pre-conditions and conditions are studied for cooperation of such units in a hierarchical organisation, as can be assumed to be the general layout of macrostructures in the brain.

## I. INTRODUCTION

Theoretic modelling in brain theory is a highly speculative subject. However, it is necessary since it seems very unlikely to get a clear picture of this very complicated device by just analyzing the available measurements on sound and/or damaged brain parts only. As in general physics, one has to realize, that there are different levels of modelling: in physics stretching from the atomary level over atom assemblies till up to general behavioural models like kinematics and mechanics, in brain theory stretching from chemical reactions over electrical spikes and neuronal cell assembly cooperation till general human behaviour.

The research discussed in this paper is located just above the direct study of synaptic cooperation of neuronal cell assemblies as studied e. g. in /Amari 1988/. It takes into account the changes of synaptic weighting, without simulating the physical details of such changes, and makes use of a general imitation of learning situation (stimuli) - response connections for building up trainable basic control loops, which allow dynamic S-R memorization and which are themselves elements of some more complex behavioural loops. The general aim of this work is to make first steps in studying structures, preconditions and conditions for building up purposeful hierarchies and by this to generate hypothesis on reasons and

meaning behind substructures in the brain like the columnar organization of the cerebral cortex (compare e. g. /Mountcastle 1978/).

The paper is organized as follows: In Chapter II a short description is given of the basic elements for building up hierarchies, the learning control loop LERNAS and on the role of its subelement AMS, some associative memory system inspired by neuronal network considerations. Chapter III starts from certain remarks on substructures in the brain and discusses the cooperation of LERNAS-elements in hierarchies as possible imitations of substructures. Chapter IV specifies the steps taken in this paper in the direction of Chapter III and Chapter V presents the results achieved by computer simulations. Finally an outlook will be given on further investigations.

## II. LERNAS AND AMS

Since the formal neuron was introduced by /McCulloch and Pitts 1943/, various kinds of neural network models have been proposed, such as the perceptron by /Rosenblatt 1957/ the neuron equation of /Caianello 1961/, the cerebellar model articulation controller CMAC by /Albus 1972, 1975/ or the associative memory models by /Fukushima 1973/, /Kohonen 1977/ and /Amari 1977/. However, the ability of such systems to store information efficiently and to perform certain pattern recognition jobs is not adequate for survival of living creatures. So they can be only substructures in the overall brain organization; one may call them a microstructure.

Purposeful acting means a goal driven coordination of sensory information and motor actions. Although the human brain is a very complex far end solution of evolution, the authors speculated in 1978 that it might be a hierarchical combination of basic elements, which would perform in an elementary way like the human brain in total, especially since there is a high similarity in the basic needs as well as in the neuronal tissue of human beings and relatively simple creatures. This led to the design of the learning control loop LERNAS in 1981 by one of the authors - /Ersü 1984/ - on the basis of psychological findings. He transformed the statement of /Piaget 1970/, that the complete intelligent action needs three elements: "1) the question, which directs possible search actions, 2) the hypothesis, which anticipates eventual solutions, 3) the control, which selects the solution to be chosen" into the structure shown in Fig. 1, by identifying the "question" with an performance criterion for assessment of possible advantages/disadvantages of certain actions, the "hypothesis" with a predictive model of environment answers and the "control" with a control strategy which selects for known situations the best action, for unknown situations some explorative action (active learning).

In detail, Fig. 1 has to be understood in the following way: The predictive model is built up in a step by step procedure from a characterization of the actual situation at the time instant $k \bullet T_s$

$T_s$ sampling time) and the measured response of the unknown environment at time instant $(k+1)T_s$. The actual situation consists of measurements regarding the stimuli and responses of the environment at time instant $k \bullet T_s$ plus - as far as necessary for a unique characterization - of the situation-stimuli and responses at time instants $(k-1)T_s$, $(k-2)T_s \ldots$, provided by the short term memory. To reduce learning effort, the associative memory system used to store the predictive model has the ability of local generalization, that means making use of the trained response value not only for the corresponding actual situation, but also in similar situations. The assessment module generates on the basis of a given goal - a wanted environment response - with an adequate performance criterion an evaluation of possible actions through testing them with the predictive model, as far as this is already built up and gives meaningful answers. The result is stored in the control strategyAMS together with its quality: real optimal action for the actual situation or only relatively optimal action, if the testing reached the border of the known area in the predictive model of the environment. In the second case, the real action is changed in a sense of curiosity, so that by the action the known area of the predictive model is extended. By this, one reaches more and more the first case, in which the real optimal actions are known. Since the first guess for a good action in the optimization phase is given to the assessment module from the control strategy AMS - not indicated in Fig. 1 to avoid unnecessary complication - finally the planning level gets superfluous and one gets very quick optimal reactions, the checking with the planning level being necessary and helpful only to find out, whether the environment has not changed, possibly. Again the associative memory system used for the control strategy is locally generalizing to reduce the necessary training effort.

The AMS storage elements for the predictive model, and for optimized actions are a refinement and implementation for on-line application of the neuronal network model CMAC from J. Albus - see e. g. /Ersü, Militzer 1982/ -, but it could be any other locally generalizing neural network model and even a storage element based on pure mathematical considerations, as has been shown in /Militzer, Tolle 1986/.

The important property to build up an excellent capability to handle different tasks in an environment known only by some sensory information - the property which qualifies LERNAS as a possible basic structure (a "ministructure") in the nervous system of living creatures - has been proven by its application to the control of a number of technical processes, starting with empty memories for the predictive model and the control strategy storage. Details on this as well as on the mathematical equations describing LERNAS can be found in /Ersü, Mao 1983/, /Ersü, Tolle 1984/ and /Ersü, Militzer 1984/.

It should be mentioned that the concept of an explicit predictive environmental model - as used in LERNAS - is neither the only meaningful description of human job handling nor a necessary part of our basic learning element. It suffices to use a prediction whether a certain action is advantegeous to reach the actual goal or whether this is not the case. More information on such a basic element MINLERNAS, which may be used instead of LERNAS in general (however, with the penalty of some performance degradation) are given in /Ersü, Tolle 1988/.

## III. HIERARCHIES

There are a number of reasons to believe, that the brain is built up as a hierarchy of control loops, the higher levels having more and more coordinative functions. A very simple example shows the necessity in certain cases. The legs of a jumping jack can move together, only. If one wants to move them separately, one has to cut the connection, has to build up a separate controller for each leg and a coordinating controller in a hierarchically higher level to restore the possibility of coordinated movements. Actually, one can find such an evolution in the historical development of certain animals. In a more complex sense a multilevel hierarchy exists in the extrapyramidal motor system. Fig. 2 from /Albus 1979/ specifies five levels of hierarchy for motor control. It can be speculated, that hierarchical organizations are not existing in the senso-motoric level only, but also in the levels of general abstractions and thinking. E. g. /Dörner 1974/ supports this idea.

If one assumes out of these indications, that hierarchies are a fundamental element of brain structuring - the details and numbers of hierarchy-levels not being known - one has to look for certain substructures and groupings of substructures in the brain. In this connection one finds as a first subdivision the cortical layers, but then as another more detailed subdivision the columns, cell assemblies heavily connected in the axis vertical to cortical layers and sparsely connected horizontally. /Mountcastle 1978/ defines minicolumns, which comprise in some neural tissue roughly 100 in other neural tissue roughly 250 individual cells. In addition to these minicolumns certain packages of minicolumns, consisting out of several hundreds of the minicolumns, can be located. They are called macrocolumns by /Mountcastle 1978/. Fig. 3 gives some abstraction, how such structures could be interpreted: each minicolumn is considered to be a ministructure of the type LERNAS, a number of LERNAS units - here shown in a ring structure instead of a filled up cylindrical structure - building up a macrocolumn. The signals between the LERNAS elements could be overlapping and cooperating. Minicolumns being elements of macrocolumns of a higher cortical layer - here layer j projecting to layer k - could initiate and/or coordinate this cooperation in a hierarchical sense.

Such a complex system is difficult to simulate. One has to go into this direction in a step by step procedure. In a first step the

overlapping or crosstalk between the minicolumns may be suppressed and the number of ministructures LERNAS representing the minicolumns should be reduced heavily. This motivates Fig. 4 as a fundamental blockdiagram for research on cooperation of LERNAS elements.

## IV. TOPICS ADDRESSED

From Fig. 4 only the lowest level of coordination (layer 1), that means the coordination of two subprocesses was implemented up to now - right half of Fig. 5. This has two reasons: Firstly, a number of fundamental questions can be posed and discussed with such a formulation already. Secondly, it is difficult to set up meaningful subprocesses and coordination goals for a higher order system.

The problem discussed in the following can be understood as the coordination of two minicolumns as described in Chapter III, but also as the coordination of higher level subtasks, which may be detailed themselves by ministructures and/or systems like Fig. 4. This is indicated in the left half of Fig. 5.

Important questions regarding hierarchies of learning control loops are:

I.    What seem to be meaningful interventions from the coordinator onto the lower level systems?

II.   Is parallel learning in both levels possible or requires a meaningful learning strategy that the control of subtasks has to be learned at first before the coordination can be learned?

III.  Normally one expects, that the lower level takes care of short term requirements and the upper level of long term strategies. Is that necessary or what happens if the upper level works on nearly the same time horizon as the lower levels?

IV.   Furtheron one expects, that the upper level may look after other goals than the lower level, e. g. the lower level tries to suppress disturbances effects since the upper level tries to minimize overall energy consumption. But can such different strategies work without oscillations or destabilization of the system?

Question I can be discussed by some general arguments, for questions II-IV only indications of possible answers can be given from simulation results. This will be postponed to Chapter V.

Fig. 6 shows three possible intervention schemes from the coordinator.

By case a) an intervention into the structure or the parameters of

the sublevel (=local) controllers is meant. Since associative mappings like AMS have no parameters being directly responsible for the behaviour of the controller - as would be the case with a parametrized linear or non-linear differential equation being the description of a conventional controller - this does not make sense for the controller built up in LERNAS. However, one could consider the possibility to change parameters or even elements, that means structural terms of the performance criterion, which is responsible for the shaping of the controller. But this would require to learn anew, which takes a too long time span in general.

By case b) a distribution of work load regarding control commands is meant. The possible idea could be, that the coordinator gives control inputs to hold the long range mean value required, since the local controllers take into account fast dynamic fluctuations only. However, this has the disadvantage that the control actions of the upper level have to be included into the inputs to the local controllers, extending the dimension of in-put space of these storage devices, since otherwise the process appears to be highly time variant for the local controllers, which is difficult to handle for LERNAS.

So case c) seems to be the best solution. In this case the coordinator commands the set points of the local controllers, generating by this local subgoals for the lower level controllers. Since this requires no input space extension for the local controllers and is in full agreement with the working conditions of single LERNAS loops, it is a meaningful and effective approach.

Fig. 7 shows the accordingly built up structure in detail. The control strategy of Fig. 1 is divided here in two parts the storage element (the controller C) and the active learning AL. The elements are explicitly characterized for the upper level only. The whole lower level is considered by the coordinator as a single pseudo-process to be controlled (see Fig. 4).

## V. SIMULATION RESULTS

For answering questions II and III the very simple non-linear process shown in Fig. 8 - detailing the subprocesses SP1, SP2 and their coupling in Fig. 7 - was used. For the comparison of bottom up and parallel learning suitably fixed PI-controllers were used for bottom up learning instead of LERNAS 1 and LERNAS 2, simulating optimally trained local controllers. Fig. 9a shows the result due to which in the first run a certain time is required for achieving a good set point following through coordinator assistance. However, with the third repetition (4th run) a good performance is reached from the first set point change on already. For parallel learning all (and not only the coordinator AMS-memories) were empty in the beginning. Practically the same performance was achieved as in bottom up training - Fig. 9b -, indicating, that at least in simple problems, as considered here, parallel learning is a real possibi-

lity. However - what is not illustrated here - the coordinator sampling time must be sufficiently long, so that the local controllers can reach the defined subgoals at least qualitatively in this time span.

For answering question III, in which respect a higher difference in the time horizon between local controller and coordinator changes the picture, a doubling of the sampling rate for the coordinator was implemented. Fig. 10 give the results. They can be interpreted as follows: Smaller sampling rates allow the coordinator to get more information about the pseudo-sub-processes, the global goal is reached faster. Larger sampling rates lead to a better overall performance when the goal is reached: there is a higher amount of averaging regarding informations about the pseudo-sub-processes.

Up to now in both levels the goal or performance criterion was the minimization of differences between the actual plant output and the requested plant output. The influence of different coordinator goals - question IV - was investigated by simulating a two stage waste water neutralization process. A detailed description of this process set up and the simulation results shall not be given here out of space reasons. It was found that:

- in hierarchical systems satisfactory overall behaviour may be reached by well defined subgoals with clearly different coordinator goals.

- since learning is goal driven, one has to accept that implicit wishes on closed loop behaviour are fulfilled by chance only. Therefore important requirements have to be included in the performance criteria explicitly.

It should be remarked finally, that one has to keep in mind, that simulation results with one single process are indications of possible behaviour only, not excluding that in other cases a fundamentally different behaviour can be met.

## VI. OUTLOOK

As has been mentioned already in Chapter III and IV, this work is one of many first steps of investigations regarding hierarchical organization in the brain, its preconditions and possible behaviour.

Subjects of further research should be the self-organizing task distribution between the processing units of each layer, and the formation of interlayer projections in order to build up meta-tasks composed of a sequence of frequently occuring elementary tasks. These investigations will on the other hand show to what extent this kind of higher-learning functions can be achieved by a hierarchy of LERNAS-type structures which model more or less low-level basic learning behaviour.

## VII. ACKNOWLEDGEMENTS

The work presented has been supported partly by the Stiftung Volkswagenwerk. The detailed evaluations of Chapter IV and V have been performed by Dipl.-Ing. M. Zoll and Dipl.-Ing. S. Gehlen. We are very thankful for this assistance.

## VIII. REFERENCES

Albus, J. S.        Theoretical and Experimental Aspects of a Cerebellar Model, Ph.D. Thesis, Univ. of Maryland, 1972

Albus, J. S.        A New Approach to Manipulator Control: The Cerebellar Model Articulation Controller (CMAC), Trans. ASmE series, G, 1975

Albus, J. S.        A Model of the Brain for Robot Control – Part 3: A Comparison of the Brain and Our Model, Byte, 1979

Amari, S. I.       Neural Theory of Association and Concept Formation, Biol. Cybernetics, Vol. 26, 1977

Amari, S. I.       Mathematical Theory of Self-Organization in Neural Nets, in: Organization of Neural Networks, Structures and Models, ed. by von Seelen, Shaw, Leinhos, VHC-Verlagsges. Weinheim, W.-Germany, 1988

Caianello, E. R.   Outline of a Theory of Thought Process and Thinking Machines, Journal of Theoretical Biology, Vol. 1, 1961

Dörner, D.         Problemlösen als Informationsverarbeitung Verlag H. Huber, 1974

Ersü, E.           On the Application of Associative Neural Network Models to Technical Control Problems, in: Localization and Orientation in Biology and Engineering, ed. by Varju, Schnitzler, Springer Verlag Berlin, W.-Germany, 1984

Ersü, E.
Mao, X.           Control of pH by Use of a Self-Organizing Concept with Associative Memories, ACI'83, Kopenhagen (Denmark), 1983

| | |
|---|---|
| Ersü, E.<br>Militzer, J. | Software Implementation of a Neuron-Like Associative Memory System for Control Application, Proceedings of the 2nd IASTED Conference on Mini- and Microcomputer Applications, MIMI'82, Davos (Switzerland), 1982 |
| Ersü, E.<br>Militzer, J. | Real-Time Implementation of an Associative Memory-Based Learning Control Scheme for Non-Linear Multivariable Processes, Symposium "Applications of Multivariable System Techniques", Plymouth (UK), 1984 |
| Ersü, E.<br>Tolle, H. | A New Concept for Learning Control Inspired by Brain Theory, Proceed. 9th IFAC World Congress, Budapest (Hungary), 1984 |
| Ersü, E.<br>Tolle, H. | Learning Control Structures with Neuron-Like Associative Memory Systems, in: Organization of Neural Networks, Structures and Models, ed. by von Seelen, Shaw, Leinhos, VCH Verlagsgesellschaft Weinheim, W.-Germany, 1988 |
| Fukushima, K. | A Model of Associative Memory in the Brain Biol. Cybernetics, Vol. 12, 1973 |
| Kohonen, T. | Associative Memory, Springer Verlag Berlin, W.-Germany, 1977 |
| McCulloch, W. S.<br>Pitts, W. H. | A Logical Calculus of the Ideas, Immanent in Nervous Activity, Bull. Math. Biophys. 9, 1943 |
| Militzer, J.<br>Tolle, H. | Vertiefungen zu einem Teilbereiche der menschlichen Intelligenz imitierenden Regelungsansatz Tagungsband-DGLR-Jahrestagung, München, W.-Germany, 1986 |
| Mountcastle, V. B. | An Organizing Principle for Cerebral Function: The Unit Module and the Distributed System, in: The Mindful Brain by G. M. Edelman, V. B. Mountcastle, The MIT-Press, Cambridge, USA, 1978 |
| Piaget, J. | Psychologie der Intelligenz, Rascher Verlag, 4th printing, 1970 |
| Rosenblatt, F. | The Perceptron: A Perceiving and Recognizing Automation, Cornell Aeronautical Laboratory, Report No. 85-460-1, 1957 |

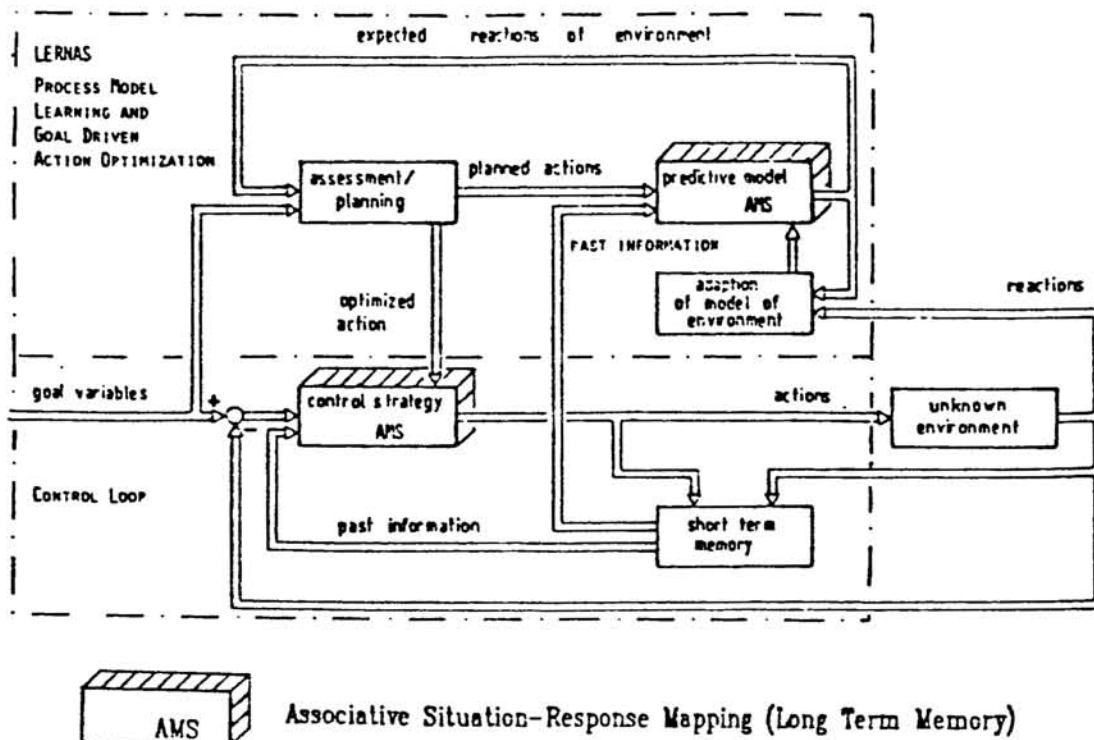

AMS — Associative Situation-Response Mapping (Long Term Memory)

Fig. 1. Architectural element LERNAS

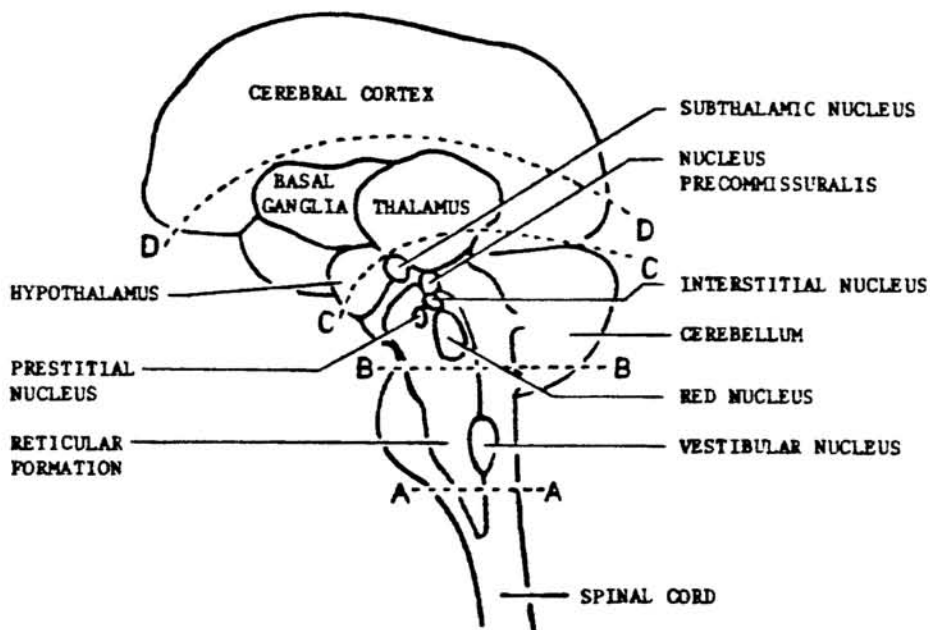

Fig. 2. The hierarchy of motor control that exists in the extra-pyramidal motor system. Basic reflexes remain even if the brain stem is cut at A-A. Coordination of these reflexes for standing is possible if the cut is at B-B. The sequential coordination required for walking requires the area below C-C to be operable. Simple tasks can be executed if the region below D-D is intact. Lengthy tasks and complex goals require the cerebral cortex. (/Albus 1979/)

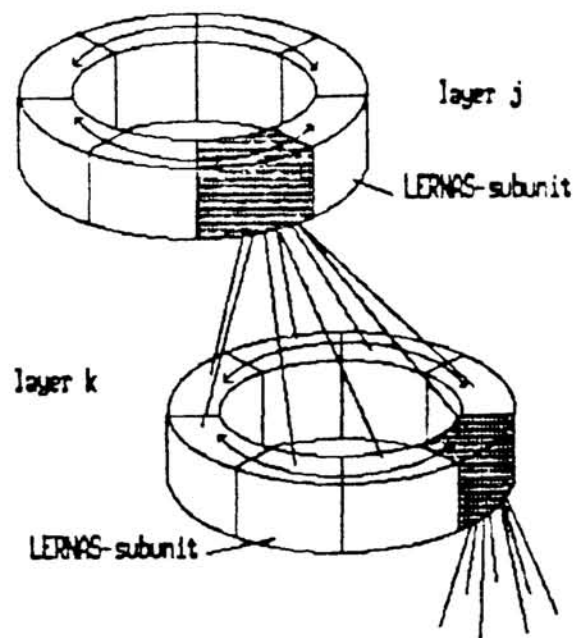

Fig. 3. Generic scetch of macrocolumns - drawn as ring
structures - from different cortical layers with
LERNAS-subunits representing minicolumns

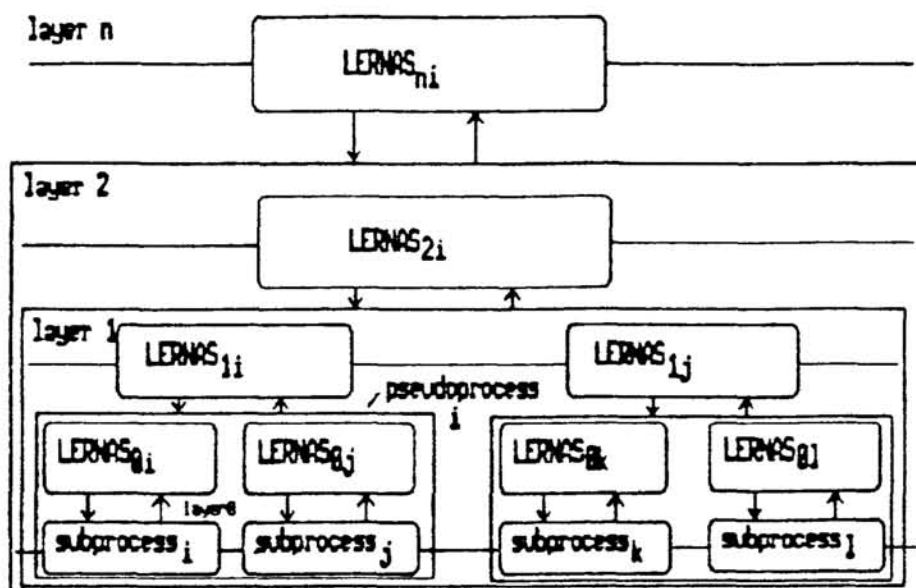

Fig. 4. LERNAS-hierarchy as a simplified research model
for cooperation of columnar structures

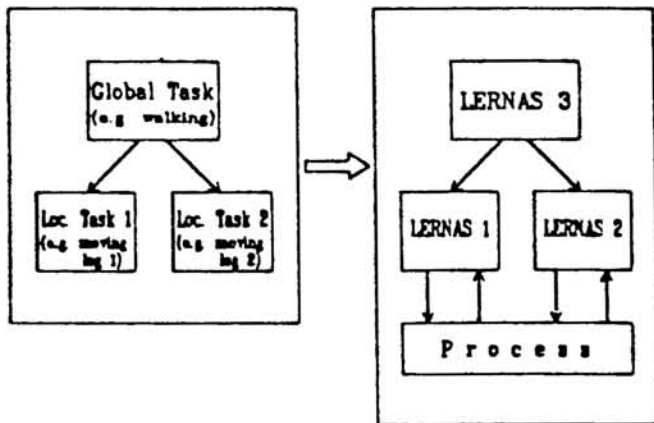 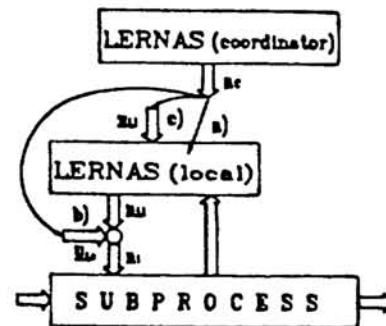

Fig. 5. Hierarchical work/
control distribution

Fig. 6. Methods of
intervention from the
coordinator

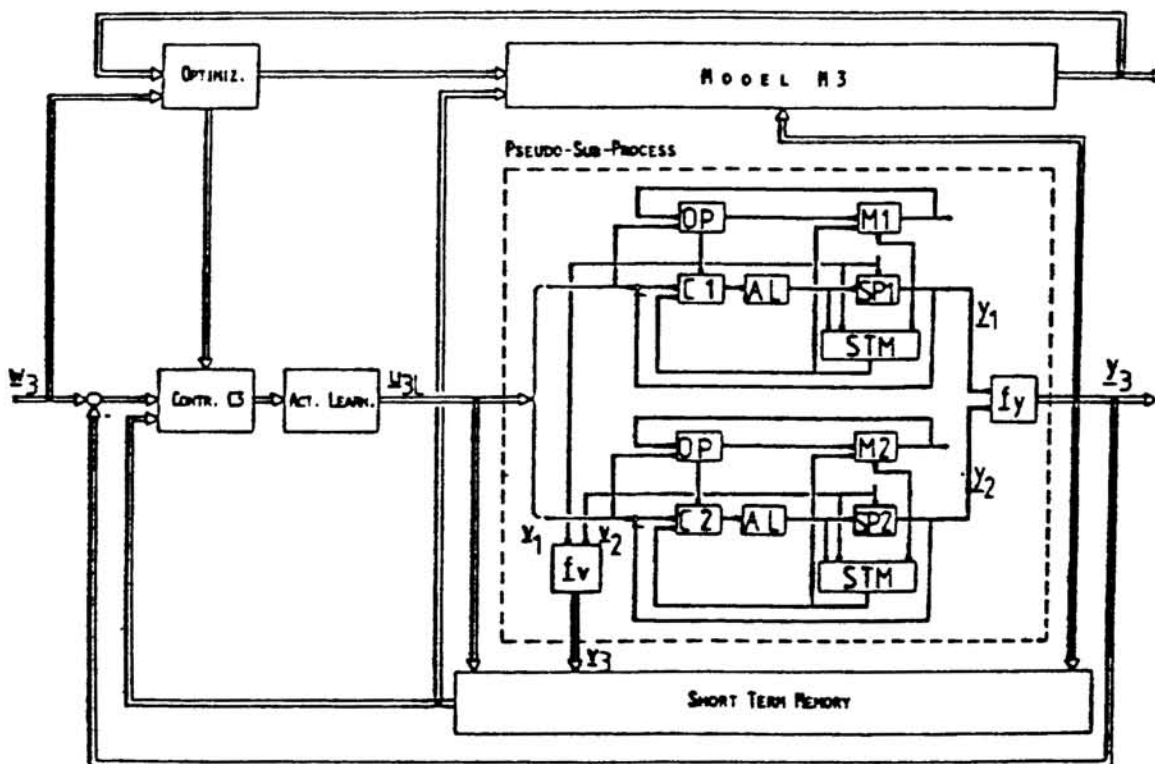

Fig. 7. Implementation of the hierarchical structure

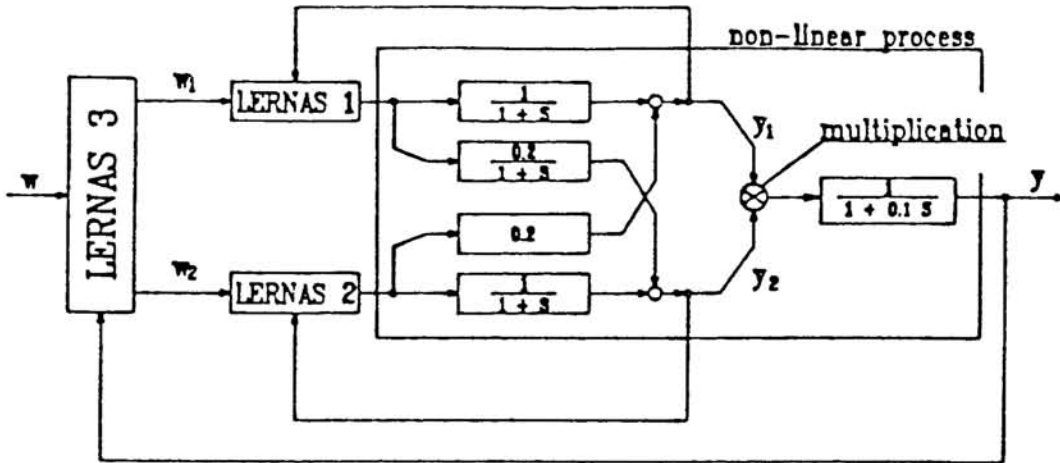

Fig. 8. Hierarchical structure with non-linear
       multivariable test-process

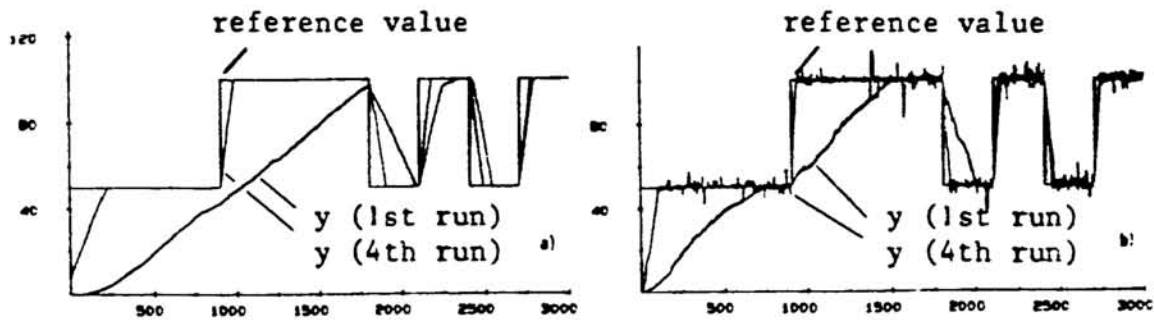

Fig. 9. Learning on coordinator level using already
       trained (a) and untrained (b) lower levels
       ($T_{coord}$ = 2 sec, $T_{loc}$ = 0.5 sec)

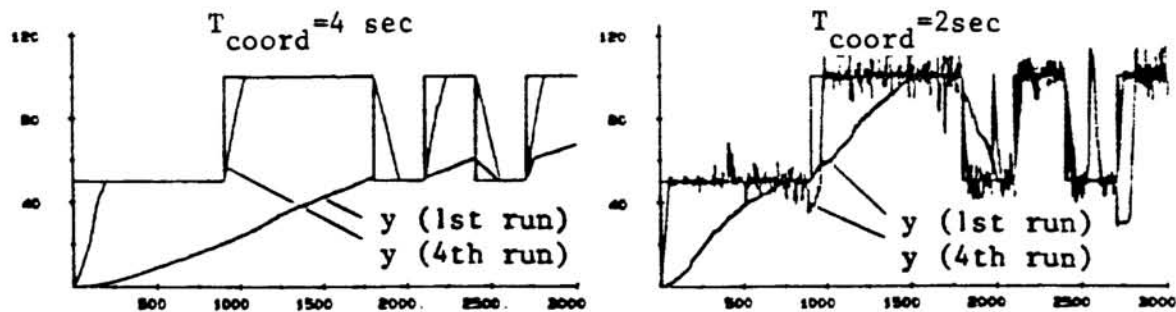

Fig. 10. Coordinator learning behaviour using different
        coordinator horizons ($T_{loc}$ = 0.5 sec)